# Convex Relaxations of Latent Variable Training

**Yuhong Guo** and **Dale Schuurmans**
Department of Computing Science
University of Alberta
{yuhong, dale}@cs.ualberta.ca

## Abstract

We investigate a new, convex relaxation of an expectation-maximization (EM) variant that approximates a standard objective while eliminating local minima. First, a cautionary result is presented, showing that any convex relaxation of EM over hidden variables must give trivial results if any dependence on the missing values is retained. Although this appears to be a strong negative outcome, we then demonstrate how the problem can be bypassed by using equivalence relations instead of value assignments over hidden variables. In particular, we develop new algorithms for estimating exponential conditional models that only require equivalence relation information over the variable values. This reformulation leads to an exact expression for EM variants in a wide range of problems. We then develop a semidefinite relaxation that yields global training by eliminating local minima.

## 1 Introduction

Few algorithms are better known in machine learning and statistics than expectation-maximization (EM) [5]. One reason is that EM solves a common problem—learning from incomplete data—that occurs in almost every area of applied statistics. Equally well known to the algorithm itself, however, is the fact that EM suffers from shortcomings. Here it is important to distinguish between the EM algorithm (essentially a coordinate descent procedure [10]) and the objective it optimizes (marginal observed or conditional hidden likelihood). Only one problem is due to the algorithm itself: since it is a simple coordinate descent, EM suffers from slow (linear) convergence and therefore can require a large number of iterations to reach a solution. Standard optimization algorithms such as quasi-Newton methods can, in principle, require exponentially fewer iterations to achieve the same accuracy (once close enough to a well behaved solution) [2, 11]. Nevertheless, EM converges quickly in many circumstances [12, 13]. The main problems attributed to EM are not problems with the algorithm per se, but instead are properties of the objective it optimizes. In particular, the standard objectives tackled by EM are not convex in any standard probability model (e.g. the exponential family). Non-convexity immediately creates the risk of local minima, which unfortunately is not just a theoretical concern: EM often does not produce very good results in practice, and can sometimes fail to improve significantly upon initial parameter settings [9]. For example, the field of unsupervised grammar induction [8] has been thwarted in its attempts to use EM for decades and is still unable to infer useful syntactic models of natural language from raw unlabeled text.

We present a convex relaxation of EM for a standard training criterion and a general class of models in an attempt to understand whether local minima are really a necessary aspect of unsupervised learning. Convex relaxations have been a popular topic in machine learning recently [4, 16]. In this paper, we propose a convex relaxation of EM that can be applied to a general class of directed graphical models, including mixture models and Bayesian networks, in the presence of hidden variables. There are some technical barriers to overcome in achieving an effective convex relaxation however. First, as we will show, *any* convex relaxation of EM must produce trivial results if it maintains any dependence on the values of hidden variables. Although this result suggests that any convex relaxation of EM cannot succeed, we subsequently show that the problem can be overcome by working

with equivalence relations over the values of the hidden variables, rather than the missing values themselves. Although equivalence relations provide an easy way to solve the symmetry collapsing problem, they do not immediately yield a convex EM formulation, because the underlying estimation principles for directed graphical models have not been formulated in these terms. Our main technical contribution therefore is a reformulation of standard estimation principles for exponential conditional models in terms of equivalence relations on variable values, rather than the variable values themselves. Given an adequate reformulation of the core estimation principle, developing a useful convex relaxation of EM becomes possible.

## 1.1 EM Variants

Before proceeding, it is important to first clarify the precise EM variant we address. In fact, there are many EM variants that optimize different criteria. Let $\mathbf{z} = (\mathbf{x}, \mathbf{y})$ denote a complete observation, where $\mathbf{x}$ refers to the observed part of the data and $\mathbf{y}$ refers to the unobserved part; and let $\mathbf{w}$ refer to the parameters of the underlying probability model, $P(\mathbf{x}, \mathbf{y}|\mathbf{w})$. (Here we consider discrete probability distributions just for simplicity of the discussion.) *Joint* and *conditional* EM algorithms are naive "self-supervised" training procedures that alternate between inferring the values of the missing variables and optimizing the parameters of the model

$$\text{(joint EM update)} \quad \mathbf{y}^{(k+1)} = \arg\max_{\mathbf{y}} P(\mathbf{x}, \mathbf{y}|\mathbf{w}^{(k)}) \quad \mathbf{w}^{(k+1)} = \arg\max_{\mathbf{w}} P(\mathbf{x}, \mathbf{y}^{(k+1)}|\mathbf{w})$$

$$\text{(conditional EM update)} \quad \mathbf{y}^{(k+1)} = \arg\max_{\mathbf{y}} P(\mathbf{y}|\mathbf{x}, \mathbf{w}^{(k)}) \quad \mathbf{w}^{(k+1)} = \arg\max_{\mathbf{w}} P(\mathbf{y}^{(k+1)}|\mathbf{x}, \mathbf{w})$$

These are clearly coordinate descent procedures that make monotonic progress in their objectives, $P(\mathbf{x}, \mathbf{y}|\mathbf{w})$ and $P(\mathbf{y}|\mathbf{x}, \mathbf{w})$. Moreover, the criteria being optimized are in fact well motivated objectives for unsupervised training: joint EM is frequently used in statistical natural language processing (where it is referred to as "Viterbi EM" [3, 7]); the conditional form has been used in [16]. The primary problem with these iterations is not that they optimize approximate or unjustified criteria, but rather that they rapidly get stuck in poor local maxima due to the extreme updates made on $\mathbf{y}$. By far, the more common form of EM—contributing the very name expectation-maximization—is given by

$$\text{(marginal EM update)} \quad \mathbf{q}_{\mathbf{y}}^{(k+1)} = P(\mathbf{y}|\mathbf{x}, \mathbf{w}^{(k)}) \quad \mathbf{w}^{(k+1)} = \arg\max_{\mathbf{w}} \sum_{\mathbf{y}} \mathbf{q}_{\mathbf{y}}^{(k+1)} \log P(\mathbf{x}, \mathbf{y}|\mathbf{w})$$

where $\mathbf{q}_{\mathbf{y}}$ is a distribution over possible missing values. Although it is not immediately obvious what this iteration optimizes, it has long been known that it monotonically improves the *marginal* likelihood $P(\mathbf{x}|\mathbf{w})$ [5]. [10] later showed that the E-step could be generalized to $\max_{\mathbf{q}_{\mathbf{y}}} \sum_{\mathbf{y}} \mathbf{q}_{\mathbf{y}} \log \left( P(\mathbf{x}, \mathbf{y}|\mathbf{w}^{(k)})/\mathbf{q}_{\mathbf{y}} \right)$. Due to the softer $\mathbf{q}_{\mathbf{y}}$ update, the standard EM update does not as converge as rapidly to a local maximum as the joint and conditional variants; however, as a result, it tends to find better local maxima. Marginal EM has subsequently become the dominant form of EM algorithm in the literature (although joint EM is still frequently used in statistical NLP [3, 7]). Nevertheless, none of the training criteria are jointly convex in the optimization variables, thus these iterations are only guaranteed to find local maxima.

Independent of the updates, the three training criteria are not equivalent nor equally well motivated. In fact, for most applications we are more interested in acquiring an accurate conditional $P(\mathbf{y}|\mathbf{x}, \mathbf{w})$, rather than optimizing the marginal $P(\mathbf{x}|\mathbf{w})$ [16]. Of the three training criteria therefore (joint, conditional and marginal), marginal likelihood appears to be the least relevant to learning predictive models. Nevertheless, the convex relaxation techniques we propose can be applied to all three objectives. For simplicity we will focus on maximizing *joint* likelihood in this paper, since it incorporates aspects of both marginal and conditional training. Conveniently, joint and marginal EM pose nearly identical optimization problems:

$$\text{(joint EM objective)} \quad \arg\max_{\mathbf{w}} \max_{\mathbf{y}} P(\mathbf{x}, \mathbf{y}|\mathbf{w}) = \arg\max_{\mathbf{w}} \max_{\mathbf{q}_{\mathbf{y}}} \left( \sum_{\mathbf{y}} \mathbf{q}_{\mathbf{y}} \log P(\mathbf{x}, \mathbf{y}|\mathbf{w}) \right)$$

$$\text{(marg. EM objective)} \quad \arg\max_{\mathbf{w}} \sum_{\mathbf{y}} P(\mathbf{x}, \mathbf{y}|\mathbf{w}) = \arg\max_{\mathbf{w}} \max_{\mathbf{q}_{\mathbf{y}}} \left( \sum_{\mathbf{y}} \mathbf{q}_{\mathbf{y}} \log P(\mathbf{x}, \mathbf{y}|\mathbf{w}) \right) + H(\mathbf{q}_{\mathbf{y}})$$

where $\mathbf{q}_{\mathbf{y}}$ is a distribution over possible missing values [10]. Therefore, much of the analysis we provide for joint EM also applies to marginal EM, leaving only a separate convex relaxation of the entropy term that can be conducted independently. We will also primarily consider the hidden variable case and assume a fixed set of random variables $Y_1, ..., Y_\ell$ is always unobserved, and a fixed set of variables $X_{\ell+1}, ..., X_n$ is always observed. The technique remains extendable to the general missing value case however.

## 2  A Cautionary Result for Convexity

Our focus in this paper will be to develop a jointly convex relaxation to the minimization problem posed by joint EM

$$\min_{\mathbf{y}} \min_{\mathbf{w}} \ -\sum_i \log P(\mathbf{x}^i, \mathbf{y}^i | \mathbf{w}) \tag{1}$$

One obvious issue we must face is to relax the discrete constraints on the assignments $\mathbf{y}$. However, the challenge is deeper than this. In the hidden variable case—when the same variables are missing in each observation—there is a complete symmetry between the missing values. In particular, for any optimal solution $(\mathbf{y}, \mathbf{w})$ there must be other, equivalent solutions $(\mathbf{y}', \mathbf{w}')$ corresponding to a permutation of the hidden variable values. Unfortunately, this form of solution symmetry has devastating consequences for any convex relaxation: Assume one attempts to use any jointly convex relaxation $f(\mathbf{q_y}, \mathbf{w})$ of the standard loglikelihood objective (1), where the the missing variable assignment $\mathbf{y}$ has been relaxed into a continuous probabilistic assignment $\mathbf{q_y}$ (like standard EM).

**Lemma 1** *If $f$ is strictly convex and invariant to permutations of unobserved variable values, then the global minimum of $f$, $(\mathbf{q_y^*}, \mathbf{w^*})$, must satisfy $\mathbf{q_y^*} = $ uniform.*

*Proof:*  Assume $(\mathbf{q_y}, \mathbf{w})$ is a global minimum of $f$ but $\mathbf{q_y} \neq$ uniform. Then there must be some permutation of the missing values, $\Pi$, such that the alternative $(\mathbf{q_y'}, \mathbf{w}') = (\Pi(\mathbf{q_y}), \Pi(\mathbf{w}))$ satisfies $\mathbf{q_y'} \neq \mathbf{q_y}$. But by the permutation invariance of $f$, this implies $f(\mathbf{q_y}, \mathbf{w}) = f(\mathbf{q_y'}, \mathbf{w}')$. By the strict convexity of $f$, we then have $f\left(\alpha(\mathbf{q_y}, \mathbf{w}) + (1-\alpha)(\mathbf{q_y'}, \mathbf{w}')\right) < \alpha f(\mathbf{q_y}, \mathbf{w}) + (1-\alpha)f(\mathbf{q_y'}, \mathbf{w}') = f(\mathbf{q_y}, \mathbf{w})$, for $0 < \alpha < 1$, contradicting the global optimality of $f(\mathbf{q_y}, \mathbf{w})$. ∎

Therefore, any convex relaxation of (1) that uses a distribution $\mathbf{q_y}$ over missing values and does not make arbitrary distinctions can never do anything but produce a uniform distribution over the hidden variable values. (The same is true for marginal and conditional versions of EM.) Moreover, any non-strictly convex relaxation must admit the uniform distribution as a possible solution. This trivialization is perhaps the main reason why standard EM objectives have not been previously convexified. (Note that standard coordinate descent algorithms simply break the symmetry arbitrarily and descend into some local solution.) This negative result seems to imply that *no* useful convex relaxation of EM is possible in the hidden variable case. However, our key observation is that a convex relaxation expressed in terms of an equivalence relation over the missing values avoids this symmetry breaking problem. In particular, equivalence relations exactly collapse the unresolvable symmetries in this context, while still representing useful structure over the hidden assignments. Representations based on equivalence relations are a useful tool for unsupervised learning that has largely been overlooked (with some exceptions [4, 15]). Our goal in this paper, therefore, will be to reformulate standard training objectives to use only equivalence relations on hidden variable values.

## 3  Directed Graphical Models

We will derive a convex relaxation framework for a general class of probability models—namely, directed models—that includes mixture models and discrete Bayesian networks as special cases. A directed model defines a joint probability distribution over a set of random variables $Z_1, ..., Z_n$ by exploiting the chain rule of probability to decompose the joint into a product of locally normalized conditional distributions $P(\mathbf{z}|\mathbf{w}) = \prod_{j=1}^{n} P(z_j|\mathbf{z}_{\pi(j)}, \mathbf{w}_j)$. Here, $\pi(j) \subseteq \{1, ..., j-1\}$, and $\mathbf{w}_j$ is the set of parameters defining conditional distribution $j$. Furthermore, we will assume an exponential family representation for the conditional distributions

$$P(z_j|\mathbf{z}_{\pi(j)}, \mathbf{w}_j) \ = \ \exp\left(\mathbf{w}_j^\top \phi_j(z_j, \mathbf{z}_{\pi(j)}) - A(\mathbf{w}_j, \mathbf{z}_{\pi(j)})\right), \quad \text{where}$$

$$A(\mathbf{w}_j, \mathbf{z}_{\pi(j)}) \ = \ \log\left(\sum_a \exp\left(\mathbf{w}_j^\top \phi_j(a, \mathbf{z}_{\pi(j)})\right)\right)$$

and $\phi_j(z_j, \mathbf{z}_{\pi(j)})$ denotes a vector of features evaluated on the value of the child and its parents. For simplicity, we will initially restrict our discussion to discrete Bayesian networks, but then reintroduce continuous random variables later. A discrete Bayesian network is just a directed model where the conditional distributions are represented by a sparse feature vector indicating the identity of the child-parent configuration $\phi_j(z_j, \mathbf{z}_{\pi(j)}) = (...1_{(z_j=a, \mathbf{z}_{\pi(j)}=\mathbf{b})}...)^\top$. That is, there is a single indicator feature for each local configuration $(a, \mathbf{b})$.

A particularly convenient property of directed models is that the complete data likelihood decomposes into an independent sum of local loglikelihoods

$$\sum_i \log P(\mathbf{z}^i|\mathbf{w}) \quad = \quad \sum_j \sum_i \mathbf{w}_j^\top \phi_j(z_j^i, \mathbf{z}_{\pi(j)}^i) - A(\mathbf{w}_j, \mathbf{z}_{\pi(j)}^i) \tag{2}$$

Thus the problem of solving for a maximum likelihood set of parameters, given complete training data, amounts to solving a set of independent log-linear regression problems, one for each variable $Z_j$. To simplify notation, consider one of the log-linear regression problems in (2) and drop the subscript $j$. Then, using a matrix notation we can rewrite the $j$th local optimization problem as

$$\min_W \left( \sum_i A(W, \Phi_{i:}) \right) - \mathrm{tr}(\Phi W Y^\top)$$

where $W \in \mathbb{R}^{c \times v}$, $\Phi \in \{0,1\}^{t \times c}$, and $Y \in \{0,1\}^{t \times v}$, such that $t$ is the number of training examples, $v$ is the number of possible values for the child variable, $c$ is the number of possible configurations for the parent variables, and tr is the matrix trace. To explain this notation, note that $Y$ and $\Phi$ are indicator matrices that have a single 1 in each row, where $Y$ indicates the value of the child variable, and $\Phi$ indicates the specific configuration of the parent values, respectively; i.e. $Y\mathbf{1} = \mathbf{1}$ and $\Phi\mathbf{1} = \mathbf{1}$, where $\mathbf{1}$ denotes the vector of all 1s. (This matrix notation greatly streamlines the presentation below.) We also use the notation $\Phi_{i:}$ to denote the $i$th row vector in $\Phi$. Here, the log normalization factor is given by $A(W, \Phi_{i:}) = \log \sum_a \exp(\Phi_{i:} W \mathbf{1}_a)$, where $\mathbf{1}_a$ denotes a sparse vector with a single 1 in position $a$.

Below, we will consider a regularized form of the objective, and thereby work with the maximum a posteriori (MAP) form of the problem

$$\min_W \left( \sum_i A(W, \Phi_{i:}) \right) - \mathrm{tr}(\Phi W Y^\top) + \frac{\alpha}{2}\mathrm{tr}(W^\top W) \tag{3}$$

This provides the core estimation principle at the heart of Bayesian network parameter learning. However, for our purposes it suffers from a major drawback: (3) is not expressed in terms of equivalence relations between the variable values. Rather it is expressed in terms of direct indicators of specific variable values in specific examples—which will lead to a trivial outcome if we attempt any convex relaxation. Instead, we require a fundamental reformulation of (3) to remove the value dependence and replace it with a dependence only on equivalence relationships.

## 4 Log-linear Regression on Equivalence Relations

The first step in reformulating (3) in terms of equivalence relations is to derive its dual.

**Lemma 2** *An equivalent optimization problem to (3) is*

$$\max_\Theta -\mathrm{tr}(\Theta \log \Theta^\top) - \frac{1}{2\alpha}\mathrm{tr}\left((Y - \Theta)^\top \Phi \Phi^\top (Y - \Theta)\right) \quad \textit{subject to } \Theta \geq 0,\ \Theta\mathbf{1} = \mathbf{1} \tag{4}$$

*Proof:* The proof follows a standard derivation, which we sketch; see e.g. [14]. First, by considering the Fenchel conjugate of $A$ it can be shown that

$$A(W, \Phi_{i:}) \quad = \quad \max_{\Theta_{i:}} \ \mathrm{tr}(\Theta_{i:}^\top \Phi_{i:} W) - \Theta_{i:} \log \Theta_{i:}^\top \quad \textit{subject to } \Theta_{i:} \geq 0,\ \Theta_{i:}\mathbf{1} = \mathbf{1}$$

Substituting this in (3) and then invoking the strong minimax property [1] allows one to show that (3) is equivalent to

$$\max_\Theta \min_W -\mathrm{tr}(\Theta \log \Theta^\top) - \mathrm{tr}((Y - \Theta)^\top \Phi W) + \frac{\alpha}{2}\mathrm{tr}(W^\top W) \quad \textit{subject to } \Theta \geq 0,\ \Theta\mathbf{1} = \mathbf{1}$$

Finally, the inner minimization can be solved by setting $W = \frac{1}{\alpha}\Phi^\top(Y - \Theta)$, yielding (4). ∎

Interestingly, deriving the dual has already achieved part of the desired result: the parent configurations now only enter the problem through the kernel matrix $K = \Phi\Phi^\top$. For Bayesian networks this kernel matrix is in fact an equivalence relation between parent configurations: $\Phi$ is a 0-1 indicator matrix with a single 1 in each row, implying that $K_{ij} = 1$ iff $\Phi_{i:} = \Phi_{j:}$, and $K_{ij} = 0$ otherwise. But more importantly, $K$ can be re-expressed as a function of the individual equivalence relations on each of the parent variables. Let $Y^p \in \{0,1\}^{t \times v_p}$ indicate the value of a parent variable $Z_p$ for each

training example. That is, $Y_{i:}^p$ is a $1 \times v_p$ sparse row vector with a single 1 indicating the value of variable $Z_p$ in example $i$. Then $M^p = Y^p Y^{p\top}$ defines an equivalence relation over the assignments to variable $Z_p$, since $M_{ij}^p = 1$ if $Y_{i:}^p = Y_{j:}^p$ and $M_{ij}^p = 0$ otherwise. It is not hard to see that the equivalence relation over complete parent configurations, $K = \Phi\Phi^\top$, is equal to the component-wise (Hadamard) product of the individual equivalence relations for each parent variable. That is, $K = \Phi\Phi^\top = M^1 \circ M^2 \circ \cdots \circ M^p$, since $K_{ij} = 1$ iff $M_{ij}^1 = 1$ and $M_{ij}^2 = 1$ and ... $M_{ij}^p = 1$.

Unfortunately, the dual problem (4) is still expressed in terms of the indicator matrix $Y$ over child variable values, which is still not acceptable. We still need to reformulate (4) in terms of the equivalence relation matrix $M = YY^\top$. Consider an alternative dual parameterization $\Omega \in \mathbb{R}^{t \times t}$ such that $\Omega \geq 0$, $\Omega\mathbf{1} = \mathbf{1}$, and $\Omega Y = \Theta$. (Note that $\Theta \in \mathbb{R}^{t \times v}$, for $v < t$, and therefore $\Omega$ is larger than $\Theta$. Also note that as long as every child value occurs at least once in the training set, $Y$ has full rank $v$. If not, then the child variable effectively has fewer values, and we could simply reduce $Y$ until it becomes full rank again without affecting the objective (3).) Therefore, since $Y$ is full rank, for any $\Theta$, some $\Omega$ must exist that achieves $\Omega Y = \Theta$. Then we can relate the primal parameters to this larger set of dual parameters by the relation $W = \frac{1}{\alpha}\Phi^\top(I - \Omega)Y$. (Even though $\Omega$ is larger than $\Theta$, they can only express the same realizable set of parameters $W$.) To simplify notation, let $B = I - \Omega$ and note the relation $W = \frac{1}{\alpha}\Phi^\top BY$. If we reparameterize the original problem using this relation, then it is possible to show that an equivalent optimization problem to (3) is given by

$$\min_B \left( \sum_i A(B, \Phi_{i:}) - \text{tr}(KBM) + \frac{1}{2\alpha}\text{tr}(B^\top KBM) \right) \quad \text{subject to } B \leq I,\, B\mathbf{1} = 0 \quad (5)$$

where $K = \Phi\Phi^\top$ and $M = YY^\top$ are equivalence relations on the parent configurations and child values respectively. The formulation (5) is now almost completely expressed in terms of equivalence relations over the data, except for one subtle problem: the log normalization factor $A(B, \Phi_{i:}) = \log \sum_a \exp\left(\frac{1}{\alpha}\Phi_{i:}\Phi^\top BY\mathbf{1}_a\right)$ still directly depends on the label indicator matrix $Y$. Our key technical lemma is that this log normalization factor can be re-expressed to depend on the equivalence relation matrix $M$ alone.

**Lemma 3** $A(B, \Phi_{i:}) = \log \sum_j \exp\left(\frac{1}{\alpha}K_{i:}BM_{:j} - \log \mathbf{1}^\top M_{:j}\right)$

*Proof:* The main observation is that an equivalence relation over value indicators, $M = YY^\top$, consists of columns copied from $Y$. That is, for all $j$, $M_{:j} = Y_{:a}$ for some $a$ corresponding to the child value in example $j$. Let $y(j)$ denote the child value in example $j$ and let $\boldsymbol{\beta}_{i:} = \frac{1}{\alpha}K_{i:}B$. Then

$$\sum_a \exp\left(\frac{1}{\alpha}\Phi_{i:}\Phi^\top BY\mathbf{1}_a\right) = \sum_a \exp(\boldsymbol{\beta}_{i:}Y_{:a}) = \sum_a \sum_{j:y(j)=a} \frac{1}{|\{\ell:y(\ell)=a\}|}\exp(\boldsymbol{\beta}_{i:}M_{:j})$$

$$= \sum_j \frac{1}{|\{\ell:y(\ell)=y(j)\}|}\exp(\boldsymbol{\beta}_{i:}M_{:j}) = \sum_j \frac{1}{\mathbf{1}^\top M_{:j}}\exp(\boldsymbol{\beta}_{i:}M_{:j}) = \sum_j \exp(\boldsymbol{\beta}_{i:}M_{:j} - \log\mathbf{1}^\top M_{:j}) \quad \blacksquare$$

Using Lemma 3 one can show that the dual problem to (5) is given by the following.

**Theorem 1** *An equivalent optimization problem to (3) is*

$$\max_{\Lambda \geq 0, \Lambda\mathbf{1}=\mathbf{1}} -tr(\Lambda \log \Lambda^\top) - \mathbf{1}^\top \Lambda \log(M\mathbf{1}) - \frac{1}{2\alpha}tr((I - \Lambda)^\top K(I - \Lambda)M) \quad (6)$$

*where $K = M^1 \circ \cdots \circ M^p$ for parent variables $Z_1, ..., Z_p$.*

*Proof:* This follows the same derivation as Lemma 2, modified by taking into account the extra term introduced by Lemma 3. First, considering the Fenchel conjugate of $A$, it can be shown that

$$A(B, \Phi_{i:}) = \max_{\Lambda_{i:} \geq 0, \Lambda_{i:}\mathbf{1}=\mathbf{1}} \frac{1}{\alpha}K_{i:}BM\Lambda_{i:}^\top - \Lambda_{i:}\log\Lambda_{i:}^\top - \Lambda_{i:}\log(M\mathbf{1})$$

Substituting this in (5) and then invoking the strong minimax property [1] allows one to show that (5) is equivalent to

$$\max_{\Lambda \geq 0, \Lambda\mathbf{1}=\mathbf{1}} \min_{B \leq I, B\mathbf{1}=0} -tr(\Lambda \log \Lambda^\top) - \mathbf{1}^\top \Lambda \log(M\mathbf{1}) - \frac{1}{\alpha}tr((I - \Lambda)^\top KBM) + \frac{1}{2\alpha}tr(B^\top KBM)$$

Finally, the inner minimization on $B$ can be solved by setting $B = I - \Lambda$, yielding (6). $\blacksquare$

This gives our key result: the log-linear regression (3) is equivalent to (6), which is now expressed strictly in terms of equivalence relations over the parent configurations and child values. That is, the value indicators, $\Phi$ and $Y$, have been successfully eliminated from the formulation. Given a solution $\Lambda^*$ to (6), the optimal model parameters $W^*$ for (3) can be recovered via $W^* = \frac{1}{\alpha}\Phi^\top(I - \Lambda^*)Y$.

# 5 Convex Relaxation of Joint EM

The equivalence relation form of log-linear regression can be used to derive useful relaxations of EM variants for directed models. In particular, by exploiting Theorem 1, we can now re-express the regularized form of the joint EM objective (1) strictly in terms of equivalence relations over the hidden variable values

$$\min_{\{Y^h\}} \sum_j \min_{\mathbf{w}_j} -\log P(z_j^i|\mathbf{z}_{\pi(j)}^i, \mathbf{w}_j) + \frac{\alpha}{2}\mathbf{w}_j^\top \mathbf{w}_j \tag{7}$$

$$= \min_{\{M^h\}} \sum_j \max_{\Lambda_j \geq 0, \Lambda_j \mathbf{1} = \mathbf{1}} -\mathrm{tr}(\Lambda_j \log \Lambda_j^\top) - \mathbf{1}^\top \Lambda_j \log(M^j \mathbf{1}) - \frac{1}{2\alpha}\mathrm{tr}\left((I - \Lambda_j)^\top K^j (I - \Lambda_j)M^j\right) \tag{8}$$

$$\text{subject to } M^h = Y^h Y^{h\top}, \, Y^h \in \{0,1\}^{t \times v_h}, \, Y^h \mathbf{1} = \mathbf{1} \tag{9}$$

where $h$ ranges over the hidden variables, and $K^j = M^{j_1} \circ \cdots \circ M^{j_p}$ for the parent variables $Z_{j_1}, ..., Z_{j_p}$ of $Z_j$.

Note that (8) is an *exact* reformulation of the joint EM objective (7); no relaxation has yet been introduced. Another nice property of the objective in (8) is that is it *concave* in each $\Lambda_j$ and *convex* in each $M^h$ individually (a maximum of convex functions is convex [2]). Therefore, (8) appears as though it might admit an efficient algorithmic solution. However, one difficulty in solving the resulting optimization problem is the constraints. Although the constraints imposed in (9) are not convex, there is a natural convex relaxation suggested by the following.

**Lemma 4** *(9) is equivalent to:* $M \in \{0,1\}^{t \times t}, diag(M) = \mathbf{1}, M = M^\top, M \succeq 0, rank(M) = v.$

A natural convex relaxation of (9) can therefore be obtained by relaxing the discreteness constraint and dropping the nonconvex rank constraint, yielding

$$M^h \in [0,1]^{t \times t}, \mathrm{diag}(M^h) = \mathbf{1}, M^h = M^{h\top}, M^h \succeq 0 \tag{10}$$

Optimizing the exact objective in (8) subject to the relaxed convex constraints (10) provides the foundation for our approach to convexifying EM. Note that since (8) and (10) are expressed solely in terms of equivalence relations, and do not depend on the specific values of hidden variables in any way, this formulation is not subject to the triviality result of Lemma 1.

However, there are still some details left to consider. First, if there is only a single hidden variable then (8) is convex with respect to the single matrix variable $M^h$. This result immediately provides a convex EM training algorithm for various applications, such as for mixture models for example (see the note regarding continuous random variables below). Second, if there are multiple hidden variables that are separated from each other (none are neighbors, nor share a common child) then the formulation (8) remains convex and can be directly applied. On the other hand, if hidden variables are connected in any way, either by sharing a parent-child relationship or having a common child, then (8) is no longer jointly convex because the trace term is no longer linear in the matrix variables $\{M^h\}$. In this case, we can restore convexity by further relaxing the problem: To illustrate, if there are multiple hidden parents $Z_{p_1}, ..., Z_{p_\ell}$ for a given child, then the combined equivalence relation $M^{p_1} \circ \cdots \circ M^{p_\ell}$ is a Hadamard product of the individual matrices. A convex formulation can be recovered by introducing an auxiliary matrix variable $\tilde{M}$ to replace $M^{p_1} \circ \cdots \circ M^{p_\ell}$ in (8) and adding the set of linear constraints $\tilde{M}_{ij} \leq M_{ij}^p$ for $p \in \{p_1, ..., p_\ell\}$, $\tilde{M}_{ij} \geq M_{ij}^{p_1} + \cdots + M_{ij}^{p_\ell} - \ell + 1$ to approximate the componentwise 'and'. A similar relaxation can also be applied when a child is hidden concurrently with hidden parent variables.

**Continuous Variables** The formulation in (8) can be applied to directed models with continuous random variables, provided that all hidden variables remain discrete. If every continuous random variable is observed, then the subproblems on these variables can be kept in their natural formulations, and hence still solved. This extension is sufficient to allow the formulation to handle Gaussian mixture models, for example. Unfortunately, the techniques developed in this paper do not apply to the situation where there are continuous hidden variables.

**Recovering the Model Parameters** Once the relaxed equivalence relation matrices $\{M^h\}$ have been obtained, the parameters of they underlying probability model need to be recovered. At an

| Bayesian networks | Fully Supervised | | Viterbi EM | | Convex EM | |
|---|---|---|---|---|---|---|
| | Train | Test | Train | Test | Train | Test |
| Synth1 | $7.23_{\pm.06}$ | $7.90_{\pm.04}$ | $11.29_{\pm.44}$ | $11.73_{\pm.38}$ | $8.96_{\pm.24}$ | $9.16_{\pm.21}$ |
| Synth2 | $4.24_{\pm.04}$ | $4.50_{\pm.03}$ | $6.02_{\pm.20}$ | $6.41_{\pm.23}$ | $5.27_{\pm.18}$ | $5.55_{\pm.19}$ |
| Synth3 | $4.93_{\pm.02}$ | $5.32_{\pm.05}$ | $7.81_{\pm.35}$ | $8.18_{\pm.33}$ | $6.23_{\pm.18}$ | $6.41_{\pm.14}$ |
| Diabetes | $5.23_{\pm.04}$ | $5.53_{\pm.04}$ | $6.70_{\pm.27}$ | $7.07_{\pm.23}$ | $6.51_{\pm.35}$ | $6.50_{\pm.28}$ |
| Pima | $5.07_{\pm.03}$ | $5.32_{\pm.03}$ | $6.74_{\pm.34}$ | $6.93_{\pm.21}$ | $5.81_{\pm.07}$ | $6.03_{\pm.09}$ |
| Cancer | $2.18_{\pm.05}$ | $2.31_{\pm.02}$ | $3.90_{\pm.31}$ | $3.94_{\pm.29}$ | $2.98_{\pm.19}$ | $3.06_{\pm.16}$ |
| Alarm | $10.23_{\pm.16}$ | $12.30_{\pm.06}$ | $11.94_{\pm.32}$ | $13.75_{\pm.17}$ | $11.74_{\pm.25}$ | $13.62_{\pm.20}$ |
| Asian | $2.17_{\pm.05}$ | $2.33_{\pm.02}$ | $2.21_{\pm.05}$ | $2.36_{\pm.03}$ | $2.70_{\pm.14}$ | $2.78_{\pm.12}$ |

Table 1: Results on synthetic and real-world Bayesian networks: average loss $\pm$ standard deviation

optimal solution to (8), one not only obtains $\{M^h\}$, but also the associated set of dual parameters $\{\Lambda_j\}$. Therefore, we can recover the primal parameters $W_j$ from the dual parameters $\Lambda_j$ by using the relationship $W_j = \frac{1}{\alpha}\Phi_j^\top(I-\Lambda_j)Y^j$ established above, which only requires availability of a label assignment matrix $Y^j$. For observed variables, $Y^j$ is known, and therefore the model parameters can be immediately recovered. For hidden variables, we first need to compute a rank $v_h$ factorization of $M^h$. Let $V = U\Sigma^{1/2}$ where $U$ and $\Sigma$ are the top $v_h$ eigenvector and eigenvalue matrices of the centered matrix $HM^hH$, such that $H = I - \frac{1}{t}\mathbf{1}\mathbf{1}^\top$. One simple idea to recover $\hat{Y}_h$ from $V$ is to run k-means on the rows of $V$ and construct the indicator matrix. A more elegant approach would be to use a randomized rounding scheme [6], which also produces a deterministic $\hat{Y}_h$, but provides some guarantees about how well $\hat{Y}_h\hat{Y}_h^\top$ approximates $M^h$. Note however that $V$ is an approximation of $Y^h$ where the row vectors have been re-centered on the origin in a rotated coordinate system. Therefore, a simpler approach is just to map the rows of $V$ back onto the simplex by translating the mean back to the simplex center and rotation the coordinates back into the positive orthant.

## 6 Experimental Results

An important question to ask is whether the relaxed, convex objective (8) is in fact over-relaxed, and whether important structure in the original marginal likelihood objective has been lost as a result. To investigate this question, we conducted a set of experiments to evaluate our convex approach compared to the standard Viterbi (i.e. joint) EM algorithm, and to supervised training on fully observed data. Our experiments are conducted using both synthetic Bayesian networks and real networks, while measuring the trained models by their logloss produced on the fully observed training data and testing data. All the results reported in this paper are averages over 10 times repeats. The test size for the experiments is 1000, the training size is 100 without specification. For a fair comparison, we used 10 random restarts for Viterbi EM to help avoid poor local optima.

For the synthetic experiments, we constructed three Bayesian networks: (1) Bayesian network 1 (Synth1) is a three layer network with 9 variables, where the two nodes in the middle layer are picked as hidden variables; (2) Bayesian network 2 (Synth2) is a network with 6 variables and 6 edges, where a node with 2 parents and 2 children is picked as hidden variable; (3) Bayesian network 3 (Synth3) is a Naive Bayes model with 7 variables, where the parent node is selected as the hidden variable. The parameters are generated in a discriminative way to produce models with apparent causal relations between the connected nodes. We performed experiments on these three synthetic networks using varying training sizes: 50, 100 and 150. Due to space limits, we only report the results for training size 100 in Table 1. Besides these three synthetic Bayesian networks, we also ran experiments using real UCI data, where we used Naive Bayes as the model structure, and set the class variables to be hidden. The middle two rows of the Table 1 show the results on two UCI data sets.

Here we can see that the convex relaxation was successful at preserving structure in the EM objective, and in fact, generally performed much better than the Viterbi EM algorithm, particularly in the case (Synth1) where there was two hidden variables. Not surprisingly, supervised training on the complete data performed better than the EM methods, but generally demonstrated a larger gap between training and test losses than the EM methods. Similar results were obtained for both

larger and smaller training sample sizes. For the UCI experiments, the results are very similar to the synthetic networks, showing good results again for the convex EM relaxation.

Finally, we conducted additional experiments on three real world Bayesian networks: Alarm, Cancer and Asian (downloaded from http://www.norsys.com/networklibrary.html). We picked one well connected node from each model to serve as the hidden variable, and generated data by sampling from the models. Table 1 shows the experimental results for these three Bayesian networks. Here we can see that the convex EM relaxation performed well on the Cancer and Alarm networks. Since we only picked one hidden variable from the 37 variables in Alarm, it is understandable that any potential advantage for the convex approach might not be large. Nevertheless, a slight advantage is still detected here. Much weaker results are obtained on the Asian network however. We are still investigating what aspects of the problem are responsible for the poorer approximation in this case.

## 7  Conclusion

We have presented a new convex relaxation of EM that obtains generally effective results in simple experimental comparisons to a standard joint EM algorithm (Viterbi EM), on both synthetic and real problems. This new approach was facilitated by a novel reformulation of log-linear regression that refers only to equivalence relation information on the data, and thereby allows us to avoid the symmetry breaking problem that blocks naive convexification strategies from working. One shortcoming of the proposed technique however is that it cannot handle continuous hidden variables; this remains a direction for future research. In one experiment, weaker approximation quality was obtained, and this too is the subject of further investigation.

## References

[1] J. Borwein and A. Lewis. *Convex Analysis and Nonlinear Optimization*. Springer, 2000.

[2] S. Boyd and L. Vandenberghe. *Convex Optimization*. Cambridge U. Press, 2004.

[3] S. Chen. Models for grapheme-to-phoneme conversion. In *Eurospeech*, 2003.

[4] T. De Bie and N. Cristianini. Fast SDP relaxations of graph cut clustering, transduction, and other combinatorial problems. *Journal of Machine Learning Research*, 7, 2006.

[5] A. Dempster, N. Laird, and D. Rubin. Maximum likelihood from incomplete data via the EM algorithm. *Journal of the Royal Statistical Society. Series B*, 39(1):1–38, 1977.

[6] M. Goemans and D. Williamson. Improved approximation algorithms for maximum cut and satisfiability problems using semidefinite programming. *JACM*, 42(6):1115–1145, 1995.

[7] S. Goldwater and M. Johnson. Bias in learning syllable structure. In *Proc. CONLL*, 2005.

[8] D. Klein and C. Manning. Corpus-based induction of syntactic structure: Models of dependency and constituency. In *Proceedings ACL*, 2004.

[9] B. Merialdo. Tagging text with a probabilistic model. *Comput. Ling.*, 20(2):155–171, 1994.

[10] R. Neal and G. Hinton. A view of the em algorithm that justifies incremental, sparse, and other variants. In M. Jordan, editor, *Learning in Graphical Models*. Kluwer, 1998.

[11] J. Nocedal and S. Wright. *Numerical Optimization*. Springer, 1999.

[12] R. Salakhutdinov, S. Roweis, and Z. Ghahramani. Optimization with EM and expectation-conjugate-gradient. In *Proceedings ICML*, 2003.

[13] N. Srebro, G. Shakhnarovich, and S. Roweis. An investigation of computational and informational limits in gaussian mixture clustering. In *Proceedings ICML*, 2006.

[14] M. Wainwright and M. Jordan. Graphical models, exponential families, and variational inference. Technical Report TR-649, UC Berkeley, Dept. Statistics, 2003.

[15] L. Xu, J. Neufeld, B. Larson, and D. Schuurmans. Max margin clustering. In *NIPS 17*, 2004.

[16] L. Xu, D. Wilkinson, F. Southey, and D. Schuurmans. Discriminative unsupervised learning of structured predictors. In *Proceedings ICML*, 2006.
